# Reducing multiclass to binary by coupling probability estimates

**Bianca Zadrozny**
Department of Computer Science and Engineering
University of California, San Diego
La Jolla, CA 92093-0114
*zadrozny@cs.ucsd.edu*

## Abstract

This paper presents a method for obtaining class membership probability estimates for multiclass classification problems by coupling the probability estimates produced by binary classifiers. This is an extension for arbitrary code matrices of a method due to Hastie and Tibshirani for pairwise coupling of probability estimates. Experimental results with Boosted Naive Bayes show that our method produces calibrated class membership probability estimates, while having similar classification accuracy as loss-based decoding, a method for obtaining the most likely class that does not generate probability estimates.

## 1 Introduction

The two most well-known approaches for reducing a multiclass classification problem to a set of binary classification problems are known as one-against-all and all-pairs. In the one-against-all approach, we train a classifier for each of the classes using as positive examples the training examples that belong to that class, and as negatives all the other training examples. In the all-pairs approach, we train a classifier for each possible pair of classes ignoring the examples that do not belong to the classes in question.

Although these two approaches are the most obvious, Allwein *et al.* [Allwein et al., 2000] have shown that there are many other ways in which a multiclass problem can be decomposed into a number of binary classification problems. We can represent each such decomposition by a code matrix $M \in \{-1, 0, +1\}^{k \times l}$, where $k$ is the number of classes and $l$ is the number of binary classification problems. If $M(c,b) = +1$ then the examples belonging to class $c$ are considered to be positive examples for the binary classification problem $b$. Similarly, if $M(c,b) = -1$ the examples belonging to $c$ are considered to be negative examples for $b$. Finally, if $M(c,b) = 0$ the examples belonging to $c$ are not used in training a classifier for $b$.

For example, in the 3-class case, the all-pairs code matrix is

|       | $b_1$ | $b_2$ | $b_3$ |
|-------|-------|-------|-------|
| $c_1$ | $+1$  | $+1$  | $0$   |
| $c_2$ | $-1$  | $0$   | $+1$  |
| $c_3$ | $0$   | $-1$  | $-1$  |

This approach for representing the decomposition of a multiclass problem into binary prob-

lems is a generalization of the Error-Correcting Output Codes (ECOC) scheme proposed by Dieterich and Bakiri [Dieterich and Bakiri, 1995]. The ECOC scheme does not allow zeros in the code matrix, meaning that all examples are used in each binary classification problem.

Orthogonal to the problem of choosing a code matrix for reducing multiclass to binary is the problem of classifying an example given the labels assigned by each binary classifier. Given an example $x$, Allwein *et al.* [Allwein et al., 2000] first create a vector $v$ of length $l$ containing the $\{-1,+1\}$ labels assigned to $x$ by each binary classifier. Then, they compute the Hamming distance between $v$ and each row of $M$, and find the row $c$ that is closest to $v$ according to this metric. The label $c$ is then assigned to $x$. This method is called Hamming decoding.

For the case in which the binary classifiers output a score whose magnitude is a measure of confidence in the prediction, they use a loss-based decoding approach that takes into account the scores to calculate the distance between $v$ and each row of $M$, instead of using the Hamming distance. This method is called loss-based decoding. Allwein *et al.* [Allwein et al., 2000] present theoretical and experimental results indicating that this method is better than Hamming decoding.

However, both of these methods simply assign a class label to each example. They do not output class membership probability estimates $\hat{P}(C = c|X = x)$ for an example $x$. These probability estimates are important when the classification outputs are not used in isolation and must be combined with other sources of information, such as misclassification costs [Zadrozny and Elkan, 2001a] or the outputs of another classifier.

Given a code matrix $M$ and a binary classification learning algorithm that outputs probability estimates, we would like to couple the estimates given by each binary classifier in order to obtain class probability membership estimates for the multiclass problem.

Hastie and Tibshirani [Hastie and Tibshirani, 1998] describe a solution for obtaining probability estimates $\hat{P}(C = c|X = x)$ in the all-pairs case by coupling the pairwise probability estimates, which we describe in Section 2. In Section 3, we extend the method to arbitrary code matrices. In Section 4 we discuss the loss-based decoding approach in more detail and compare it mathematically to the method by Hastie and Tibshirani. In Section 5 we present experimental results.

## 2   Coupling pairwise probability estimates

We are given pairwise probability estimates $r_{ij}(x)$ for every class $i \neq j$, obtained by training a classifier using the examples belonging to class $i$ as positives and the examples belonging to class $j$ as negatives. We would like to couple these estimates to obtain a set of class membership probabilities $p_i(x) = P(C = c_i|X = x)$ for each example $x$. The $r_{ij}$ are related to the $p_i$ according to

$$r_{ij}(x) = P(C = i|C = i \vee C = j, X = x) = \frac{p_i(x)}{p_i(x) + p_j(x)}$$

Since we additionally require that $\sum_i p_i(x) = 1$, there are $k - 1$ free parameters and $k(k - 1)/2$ constraints. This implies that there may not exist $p_i$ satisfying these constraints.

Let $n_{ij}$ be the number of training examples used to train the binary classifier that predicts $r_{ij}$. In order to find the best approximation $\hat{r}_{ij}(x) = \hat{p}_i(x)/(\hat{p}_i(x) + \hat{p}_j(x))$, Hastie and Tibshirani fit the Bradley-Terrey model for paired comparisons [Bradley and Terry, 1952] by minimizing the average weighted Kullback-Leibler distance $l(x)$ between $r_{ij}(x)$ and

$\hat{r}_{ij}(x)$ for each $x$, given by

$$l(x) = \sum_{i \neq j} n_{ij} \left[ r_{ij}(x) \log \frac{r_{ij}(x)}{\hat{r}_j(x)} + (1 - r_{ij}(x)) \log \frac{1 - r_{ij}(x)}{1 - \hat{r}_j(x)} \right]$$

The algorithm is as follows:

1. Start with some guess for the $\hat{p}(x)$ and corresponding $\hat{r}_j(x)$.
2. Repeat until convergence:
   (a) For each $i = 1, 2, \ldots, k$

$$\hat{p}(x) \leftarrow \hat{p}(x) \frac{\sum_{j \neq i} n_{ij} r_{ij}(x)}{\sum_{j \neq i} n_{ij} \hat{r}_j(x)}$$

   (b) Renormalize the $\hat{p}(x)$.
   (c) Recompute the $\hat{r}_j(x)$.

Hastie and Tibshirani [Hastie and Tibshirani, 1998] prove that the Kullback-Leibler distance between $r_{ij}(x)$ and $\hat{r}_{ij}(x)$ decreases at each step. Since this distance is bounded below by zero, the algorithm converges. At convergence, the $\hat{r}_{ij}$ are consistent with the $\hat{p}_i$. The class predicted for each example $x$ is $\hat{c}(x) = \text{argmax } \hat{p}_i(x)$.

Hastie and Tibshirani also prove that the $\hat{p}_i(x)$ are in the same order as the non-iterative estimates $\tilde{p}_i(x) = \sum_{j \neq i} r_{ij}(x)$ for each $x$. Thus, the $\tilde{p}_i(x)$ are sufficient for predicting the most likely class for each example. However, as shown by Hastie and Tibshirani, they are not accurate probability estimates because they tend to underestimate the differences between the $\hat{p}_i(x)$ values.

## 3   Extending the Hastie-Tibshirani method to arbitrary code matrices

For an arbitrary code matrix $M$, instead of having pairwise probability estimates, we have an estimate $r_b(x)$ for each column $b$ of $M$, such that

$$r_b(x) = P(\bigvee_{c \in I} C = c \mid \bigvee_{c \in I \cup J} C = c, X = x) = \frac{\sum_{c \in I} p_c(x)}{\sum_{c \in I \cup J} p_c(x)}$$

where $I$ and $J$ are the set of classes for which $M(\cdot, b) = 1$ and $M(\cdot, b) = -1$, respectively.

We would like to obtain a set of class membership probabilities $p_i(x)$ for each example $x$ compatible with the $r_b(x)$ and subject to $\sum_i p_i(x) = 1$. In this case, the number of free parameters is $k - 1$ and the number of constraints is $l + 1$, where $l$ is the number of columns of the code matrix.

Since for most code matrices $l$ is greater than $k - 1$, in general there is no exact solution to this problem. For this reason, we propose an algorithm analogous to the Hastie-Tibshirani method presented in the previous section to find the best approximate probability estimates $\hat{p}_i(x)$ such that

$$\hat{r}_b(x) = \frac{\sum_{c \in I} \hat{p}(x)}{\sum_{c \in I \cup J} \hat{p}(x)},$$

and the Kullback-Leibler distance between $\hat{r}_b(x)$ and $r_b(x)$ is minimized.

Let $n_b$ be the number of training examples used to train the binary classifier that corresponds to column $b$ of the code matrix. The algorithm is as follows:

1. Start with some guess for the $\hat{p}(x)$ and corresponding $\hat{r}_b(x)$.
2. Repeat until convergence:

(a) For each $i = 1, 2, \ldots, k$

$$\hat{p}(x) \leftarrow \hat{p}(x) \frac{\sum_{b \; s.t. \; M(i,b)=1} n_b r_b(x) + \sum_{b \; s.t. \; M(i,b)=-1} n_b(1 - r_b(x))}{\sum_{b \; s.t. \; M(i,b)=1} n_b \hat{r}(x) + \sum_{b \; s.t. \; M(i,b)=-1} n_b(1 - \hat{r}(x))}$$

(b) Renormalize the $\hat{p}(x)$.

(c) Recompute the $\hat{r}(x)$.

If the code matrix is the all-pairs matrix, this algorithm reduces to the original method by Hastie and Tibshirani.

Let $B_{+i}$ be the set of matrix columns for which $M(i, \cdot) = +1$ and $B_{-i}$ be the set of matrix columns for which $M(c, \cdot) = -1$. By analogy with the non-iterative estimates suggested by Hastie and Tibshirani, we can define non-iterative estimates $\tilde{p}_i(x) = \sum_{b \in B_{+i}} r_b(x) + \sum_{b \in B_{-i}} (1 - r_b(x))$. For the all-pairs code matrix, these estimates are the same as the ones suggested by Hastie and Tibshirani. However, for arbitrary matrices, we cannot prove that the non-iterative estimates predict the same class as the iterative estimates.

# 4 Loss-based decoding

In this section, we discuss how to apply the loss-based decoding method to classifiers that output class membership probability estimates. We also study the conditions under which this method predicts the same class as the Hastie-Tibshirani method, in the all-pairs case.

The loss-based decoding method [Allwein et al., 2000] requires that each binary classifier output a margin score satisfying two requirements. First, the score should be positive if the example is classified as positive, and negative if the example is classified as negative. Second, the magnitude of the score should be a measure of confidence in the prediction.

The method works as follows. Let $f(x, b)$ be the margin score predicted by the classifier corresponding to column $b$ of the code matrix for example $x$. For each row $c$ of the code matrix $M$ and for each example $x$, we compute the distance between $f$ and $M(c, \cdot)$ as

$$d_L(x, c) = \sum_{b=1}^{l} L(M(c, b) f(x, b)) \tag{1}$$

where $L$ is a loss function that is dependent on the nature of the binary classifier and $M(c, b)$ = 0, 1 or −1. We then label each example $x$ with the label $c^\star$ for which $d_L$ is minimized.

If the binary classification learning algorithm outputs scores that are probability estimates, they do not satisfy the first requirement because the probability estimates are all between 0 and 1. However, we can transform the probability estimates $r_b(x)$ output by each classifier $b$ into margin scores by subtracting $1/2$ from the scores, so that we consider as positives the examples $x$ for which $r_b(x)$ is above $1/2$, and as negatives the examples $x$ for which $r_b(x)$ is below $1/2$.

We now prove a theorem that relates the loss-based decoding method to the Hastie-Tibshirani method, for a particular class of loss functions.

**Theorem 1** The loss-based decoding method for all-pairs code matrices predicts the same class label as the iterative estimates $\hat{p}_i(x)$ given by Hastie and Tibshirani, if the loss function is of the form $L(y) = -ay$, for any $a > 0$.

**Proof:** We first show that, if the loss function is of the form $L(y) = -ay$, the loss-based decoding method predicts the same class label as the non-iterative estimates $\tilde{p}_i(x)$, for the all-pairs code matrix.

| Dataset | #Training Examples | #Test Examples | #Attributes | #Classes |
|---------|--------------------|-----------------|-------------|----------|
| satimage | 4435 | 2000 | 36 | 7 |
| pendigits | 7494 | 3498 | 16 | 10 |
| soybean | 307 | 376 | 35 | 9 |

Table 1: Characteristics of the datasets used in the experiments.

The non-iterative estimates $\tilde{p}_i(x)$ are given by

$$\tilde{p}(x) = \sum_{b \in B_{+c}} r_b(x) + \sum_{b \in B_{-c}} (1 - r_b(x)) = \sum_{b \in B_{+c}} r_b(x) - \sum_{b \in B_{-c}} r_b(x) + |B_{-c}|,$$

where $B_{+c}$ and $B_{-c}$ are the sets of matrix columns for which $M(c, \cdot) = +1$ and $M(c, \cdot) = -1$, respectively.

Considering that $L(y) = -ay$ and $f(x, b) = r_b(x) - 1/2$, and eliminating the terms for which $M(c, b) = 0$, we can rewrite Equation 1 as

$$d(x, c) = \sum_{b \in B_{+c}} -a \left( r_b(x) - \frac{1}{2} \right) + \sum_{b \in B_{-c}} a \left( r_b(x) - \frac{1}{2} \right) = -a \left( \sum_{b \in B_{+c}} r_b(x) - \sum_{b \in B_{-c}} r_b(x) + \frac{1}{2} (|B_{-c}| - |B_{+c}|) \right).$$

For the all-pairs code matrix the following relationship holds: $1/2(|B_{-c}| - |B_{+c}|) = |B_{-c}| + (k-1)/2$, where $k$ is the number of classes. So, the distance $d(x, c)$ is

$$d(x, c) = -a \left( \sum_{b \in B_{+c}} r_b(x) - \sum_{b \in B_{-c}} r_b(x) + |B_{-c}| + (k-1)/2 \right).$$

It is now easy to see that the class $c^\star(x)$ which minimizes $d(x, c)$ for example $x$, also maximizes $\tilde{p}_c(x)$. Furthermore, if $d(x, i) < d(x, j)$ then $p(x, i) > p(x, j)$, which means that the ranking of the classes for each example is the same.

Since the non-iterative estimates $\tilde{p}_c(x)$ are in the same order as the iterative estimates $\hat{p}_c(x)$, we can conclude that the Hastie-Tibshirani method is equivalent to the loss-based decoding method if $L(y) = -ay$, in terms of class prediction, for the all-pairs code matrix. ∎

Allwein *et al.* do not consider loss functions of the form $L(y) = -ay$, and uses non-linear loss functions such as $L(y) = e^{-y}$. In this case, the class predicted by loss-based decoding may differ from the one predicted by the method by Hastie and Tibshirani.

This theorem applies only to the all-pairs code matrix. For other matrices such that $|B_{-c}| - |B_{+c}|$ is a linear function of $|B_{-c}|$ (such as the one-against-all matrix), we can prove that loss-based decoding (with $L(y) = -ay$) predicts the same class as the non-iterative estimates. However, in this case, the non-iterative estimates do not necessarily predict the same class as the iterative ones.

# 5 Experiments

We performed experiments using the following multiclass datasets from the UCI Machine Learning Repository [Blake and Merz, 1998]: `satimage`, `pendigits` and `soybean`. Table 1 summarizes the characteristics of each dataset.

The binary learning algorithm used in the experiments is boosted naive Bayes [Elkan, 1997], since this is a method that cannot be easily extended to handle multiclass problems directly. For all the experiments, we ran 10 rounds of boosting.

| Method | Code Matrix | Error Rate | MSE |
|---|---|---|---|
| Loss-based ($L(y) = -y$) | All-pairs | 0.1385 | - |
| Loss-based ($L(y) = e^{-y}$) | All-pairs | 0.1385 | - |
| Hastie-Tibshirani (non-iterative) | All-pairs | 0.1385 | 0.0999 |
| Hastie-Tibshirani (iterative) | All-pairs | 0.1385 | 0.0395 |
| Loss-based ($L(y) = -y$) | One-against-all | 0.1445 | - |
| Loss-based ($L(y) = e^{-y}$) | One-against-all | 0.1425 | - |
| Extended Hastie-Tibshirani (non-iterative) | One-against-all | 0.1445 | 0.1212 |
| Extended Hastie-Tibshirani (iterative) | One-against-all | 0.1670 | 0.0396 |
| Loss-based ($L(y) = -y$) | Sparse | 0.1435 | - |
| Loss-based ($L(y) = e^{-y}$) | Sparse | 0.1425 | - |
| Extended Hastie-Tibshirani (non-iterative) | Sparse | 0.1480 | 0.1085 |
| Extended Hastie-Tibshirani (iterative) | Sparse | 0.1330 | 0.0340 |
| Multiclass Naive Bayes | - | 0.2040 | 0.0651 |

Table 2: Test set results on the `satimage` dataset.

We use three different code matrices for each dataset: all-pairs, one-against-all and a sparse random matrix. The sparse random matrices have $\lceil 15 \log_2 k \rceil$ columns, and each element is 0 with probability 1/2 and -1 or +1 with probability 1/4 each. This is the same type of sparse random matrix used by Allwein *et al.*[Allwein et al., 2000]. In order to have good error correcting properties, the Hamming distance $\rho$ between each pair of rows in the matrix must be large. We select the matrix by generating 10,000 random matrices and selecting the one for which $\rho$ is maximized, checking that each column has at least one $-1$ and one $+1$, and that the matrix does not have two identical columns.

We evaluate the performance of each method using two metrics. The first metric is the error rate obtained when we assign each example to the most likely class predicted by the method. This metric is sufficient if we are only interested in classifying the examples correctly and do not need accurate probability estimates of class membership.

The second metric is squared error, defined for one example $x$ as $SE(x) = \sum_j (t_j(x) - p_j(x))^2$, where $p_j(x)$ is the probability estimated by the method for example $x$ and class $j$, and $t_j(x)$ is the true probability of class $j$ for $x$. Since for most real-world datasets true labels are known, but not probabilities, $t_j(x)$ is defined to be 1 if the label of $x$ is $j$ and 0 otherwise. We calculate the squared error for each $x$ to obtain the mean squared error (MSE). The mean squared error is an adequate metrics for assessing the accuracy of probability estimates [Zadrozny and Elkan, 2001b]. This metric cannot be applied to the loss-based decoding method, since it does not produce probability estimates.

Table 2 shows the results of the experiments on the `satimage` dataset for each type of code matrix. As a baseline for comparison, we also show the results of applying multiclass Naive Bayes to this dataset. We can see that the iterative Hastie-Tibshirani procedure (and its extension to arbitrary code matrices) succeeds in lowering the MSE significantly compared to the non-iterative estimates, which indicates that it produces probability estimates that are more accurate. In terms of error rate, the differences between methods are small. For one-against-all matrices, the iterative method performs consistently worse, while for sparse random matrices, it performs consistently better. Figure 1 shows how the MSE is lowered at each iteration of the Hastie-Tibshirani algorithm, for the three types of code matrices.

Table 3 shows the results of the same experiments on the datasets `pendigits` and `soybean`. Again, the MSE is significantly lowered by the iterative procedure, in all cases. For the `soybean` dataset, using the sparse random matrix, the iterative method again has a lower error rate than the other methods, which is even lower than the error rate using the all-pairs matrix. This is an interesting result, since in this case the all-pairs matrix has 171 columns (corresponding to 171 classifiers), while the sparse matrix has only 64 columns.

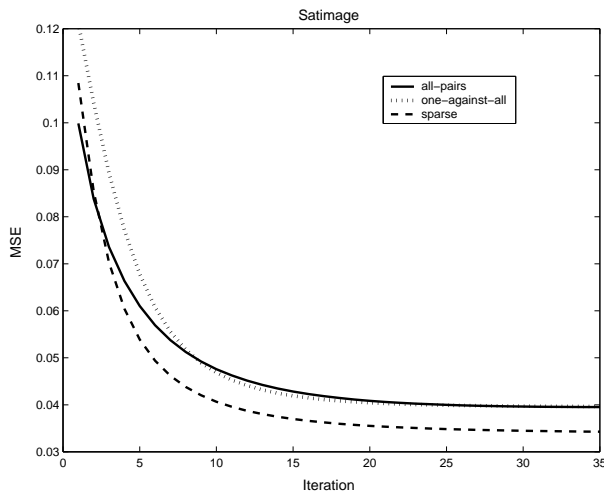

Figure 1: Convergence of the MSE for the `satimage` dataset.

| Method | Code Matrix | pendigits | | soybean | |
|---|---|---|---|---|---|
| | | Error Rate | MSE | Error Rate | MSE |
| Loss-based ($L(y) = -y$) | All-pairs | 0.0723 | - | 0.0665 | - |
| Loss-based ($L(y) = e^{-y}$) | All-pairs | 0.0715 | - | 0.0665 | - |
| Hastie-Tibshirani (non-iterative) | All-pairs | 0.0723 | 0.0747 | 0.0665 | 0.0454 |
| Hastie-Tibshirani (iterative) | All-pairs | 0.0718 | 0.0129 | 0.0665 | 0.0066 |
| Loss-based ($L(y) = -y$) | One-against-all | 0.0963 | - | 0.0824 | - |
| Loss-based ($L(y) = e^{-y}$) | One-against-all | 0.0963 | - | 0.0931 | - |
| Ext. Hastie-Tibshirani (non-it.) | One-against-all | 0.0963 | 0.0862 | 0.0824 | 0.0493 |
| Ext. Hastie-Tibshirani (it.) | One-against-all | 0.1023 | 0.0160 | 0.0931 | 0.0073 |
| Loss-based ($L(y) = -y$) | Sparse | 0.1284 | - | 0.0718 | - |
| Loss-based ($L(y) = e^{-y}$) | Sparse | 0.1266 | - | 0.0718 | - |
| Ext. Hastie-Tibshirani (non-it.) | Sparse | 0.1484 | 0.0789 | 0.0798 | 0.0463 |
| Ext. Hastie-Tibshirani (it.) | Sparse | 0.1261 | 0.0216 | 0.0636 | 0.0062 |
| Multiclass Naive Bayes | - | 0.2779 | 0.0509 | 0.0745 | 0.0996 |

Table 3: Test set results on the `pendigits` and `soybean` datasets.

## 6  Conclusions

We have presented a method for producing class membership probability estimates for multiclass problems, given probability estimates for a series of binary problems determined by an arbitrary code matrix.

Since research in designing optimal code matrices is still on-going [Utschick and Weichselberger, 2001] [Crammer and Singer, 2000], it is important to be able to obtain class membership probability estimates from arbitrary code matrices. In current research, the effectiveness of a code matrix is determined primarily by the classification accuracy. However, since many applications require accurate class membership probability estimates for each of the classes, it is important to also compare the different types of code matrices according to their ability of producing such estimates. Our extension of Hastie and Tibshirani's method is useful for this purpose.

Our method relies on the probability estimates given by the binary classifiers to produce the multiclass probability estimates. However, the probability estimates produced by Boosted

Naive Bayes are not calibrated probability estimates. An interesting direction for future work is in determining whether the calibration of the probability estimates given by the binary classifiers improves the calibration of the multiclass probabilities.

## References

[Allwein et al., 2000] Allwein, E. L., Schapire, R. E., and Singer, Y. (2000). Reducing multiclass to binary: A unifying approach for margin classifiers. *Journal of Machine Learning Research*, 1:113–141.

[Blake and Merz, 1998] Blake, C. L. and Merz, C. J. (1998). UCI repository of machine learning databases. Department of Information and Computer Sciences, University of California, Irvine. `http://www.ics.uci.edu/~mlearn/MLRepository.html`.

[Bradley and Terry, 1952] Bradley, R. and Terry, M. (1952). Rank analysis of incomplete block designs, I: The method of paired comparisons. *Biometrics*, pages 324–345.

[Crammer and Singer, 2000] Crammer, K. and Singer, Y. (2000). On the learnability and design of output codes for multiclass problems. In *Proceedings of the Thirteenth Annual Conference on Computational Learning Theory*, pages 35–46.

[Dietterich and Bakiri, 1995] Dietterich, T. G. and Bakiri, G. (1995). Solving multiclass learning problems via error-correcting output codes. *Journal of Artificial Intelligence Research*, 2:263–286.

[Elkan, 1997] Elkan, C. (1997). Boosting and naive bayesian learning. Technical Report CS97-557, University of California, San Diego.

[Hastie and Tibshirani, 1998] Hastie, T. and Tibshirani, R. (1998). Classification by pairwise coupling. In *Advances in Neural Information Processing Systems*, volume 10. MIT Press.

[Utschick and Weichselberger, 2001] Utschick, W. and Weichselberger, W. (2001). Stochastic organization of output codes in multiclass learning problems. *Neural Computation*, 13(5):1065–1102.

[Zadrozny and Elkan, 2001a] Zadrozny, B. and Elkan, C. (2001a). Learning and making decisions when costs and probabilities are both unknown. In *Proceedings of the Seventh International Conference on Knowledge Discovery and Data Mining*, pages 204–213. ACM Press.

[Zadrozny and Elkan, 2001b] Zadrozny, B. and Elkan, C. (2001b). Obtaining calibrated probability estimates from decision trees and naive bayesian classifiers. In *Proceedings of the Eighteenth International Conference on Machine Learning*, pages 609–616. Morgan Kaufmann Publishers, Inc.
